# A Quantitative Model of Counterfactual Reasoning

**Daniel Yarlett**
Division of Informatics
University of Edinburgh
Edinburgh, Scotland
*dany@cogsci.ed.ac.uk*

**Michael Ramscar**
Division of Informatics
University of Edinburgh
Edinburgh, Scotland
*michael@dai.ed.ac.uk*

## Abstract

In this paper we explore two quantitative approaches to the modelling of counterfactual reasoning – a linear and a noisy-OR model – based on information contained in conceptual dependency networks. Empirical data is acquired in a study and the fit of the models compared to it. We conclude by considering the appropriateness of non-parametric approaches to counterfactual reasoning, and examining the prospects for other parametric approaches in the future.

## 1    Introduction

If robins didn't have wings would they still be able to fly, eat worms or build nests? Previous work on counterfactual reasoning has tended to characterise the processes by which questions such as these are answered in purely qualitative terms, either focusing on the factors determining their onset and consequences (see Roese, 1997, for a review); the qualitative outline of their psychological characteristics (Kahneman and Miller, 1986; Byrne and Tasso, 1999); or else their logical or schematic properties (Lewis, 1973; Goodman, 1983). And although Pearl (2000) has described a formalism addressing quantitative aspects of counterfactual reasoning, this model has yet to be tested empirically. Furthermore, the non-parametric framework in which it is proposed means certain problems attach to it as a cognitive model, as discussed in §6.

To date then, the quantitative processes underlying human counterfactual reasoning have proven surprisingly recalcitrant to philosophical, psychological and linguistic analysis. In this paper we propose two parametric models of counterfactual reasoning for a specific class of counterfactuals: those involving modifications to our conceptual knowledge. The models we present are intended to capture the constraints operative on this form of inference at the computational level. Having outlined the models, we present a study which compares their predictions with the judgements of participants about corresponding counterfactuals. Finally, we conclude by raising logistical and methodological doubts about a non-parametric approach to the problem, and considering future work to extend the current models.

## 2 Counterfactuals and Causal Dependencies

One of the main difficulties in analysing counterfactuals is that they refer to alternative ways that things could be, but it's difficult to specify exactly which alternatives they pick out. For example, to answer the counterfactual question we began this paper with we clearly need to examine the possible states of affairs in which robins don't have wings in order to see whether they will still be able to fly, eat worms and build nests in them. But the problem is that we can imagine many possible ways in which robins can be without wings – for instance, at an extreme we can imagine a situation in which the robin genotype failed to evolve beyond the plankton stage – not all of which will be relevant when it comes to reasoning counterfactually.

In the alternatives envisaged by a counterfactual some things are clearly going to differ from the way they are in the actual world, while others are going to remain unchanged. And specifying which things will be affected, and which things will be unaffected, by a counterfactual supposition is the crux of the issue. Counterfactual reasoning seems to revolve around causal dependencies: if something depends on a counterfactual supposition then it should differ from the way it is in the actual world, otherwise it should remain just as it is. The challenge is to specify exactly what depends on what in the world – and crucially to what degree, if we are interested in the quantitative aspects of counterfactual reasoning – in order that we can arrive at appropriate counterfactual inferences. Clearly some information about our representation of dependency relations is required.

## 3 Dependency Information

Fortunately, data is available about people's representations of dependencies, albeit in a limited domain. As part of an investigation into feature centrality, Sloman, Love and Ahn (1998) explored the idea that a feature is central to a concept to the degree that other features depend on it. To test this idea empirically they derived dependency networks for four concepts – *robin*, *apple*, *chair* and *guitar* – by asking people to rate on a scale of 0 to 3 how strongly they thought the features of the four concepts depended on one another. One of the dependency structures derived from this process is depicted in Figure 1.

## 4 Parametric Models

The models we present here simulate counterfactual reasoning about a concept by operating on conceptual networks such as the one in Figure 1. A counterfactual supposition is entertained by setting the activation of the counterfactually manipulated feature to an appropriate level. Inference then proceeds via an iterative algorithm which propagates the effect of manipulating the selected feature throughout the network.

In order to do this we make two main assumptions about cause-effect interactions. First we assume that a node representing an effect, $E$, will be expected to change as a function of (i) the degree to which a node representing its cause, $C$, has itself changed, *and* (ii) the degree to which $E$ depends on $C$. Second, we also assume that multiple cause nodes, $C_{1-n}$, will affect a target node, $E$, independently of one another and in a cumulative fashion. This means that the proposed models do not attempt to deal with interactions between causes.

The first assumption seems warranted by recent empirical work (Yarlett & Ramscar, in preparation). And while the second assumption is certainly not true in all instances – interaction effects are certainly possible – there do seem to be multiple schemas that can be adopted in causal reasoning (Kelley, 1967), and it may be that the parametric assumptions of the two models correspond to a form of reasoning that predominates.

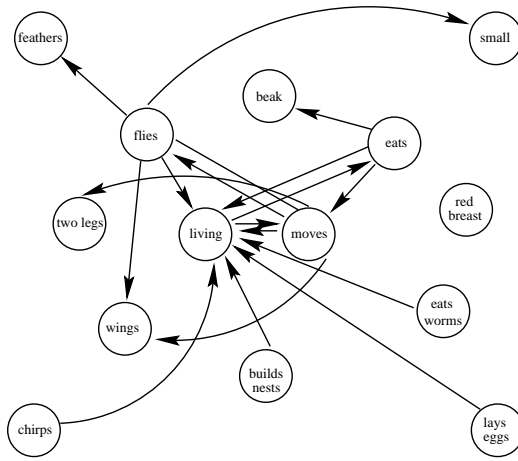

Figure 1: Dependency network for the concept *robin*. An arrow drawn from feature A to feature B means that A depends on B. Note (i) that only the strongest dependency links are shown, but that *all* dependency information was used in the simulations; (ii) there is a numeric strength associated with every dependency connection, although this is not shown in the diagram; and (iii) the proposed models propagate information in the *opposite* direction to the dependency connections.

## 4.1 Causal Dependency Networks

The dependency networks obtained by Sloman, Love and Ahn (1998) were collected by asking people to consider features in a pairwise fashion, independently of all other features. However, causal inference requires that the causal impact of multiple features on a target node be combined. Therefore some preprocessing needs to be done to the raw dependency networks to define a causal dependency network suitable for using in counterfactual inference. The original dependency networks can each be represented as a matrix $D^n$, in which $d_{ij}^n$ represents the strength with which feature $j$ depends on feature $i$ in concept $n$, as judged by the original participants. The modified causal dependency networks, $C^n$, are defined as follows:

$$c_{ij}^n = \frac{d_{ij}^n}{3 \sum_k f(d_{kj}^n)} \tag{1}$$

where

$$f(v) = \begin{cases} 1 & \text{where } v > 0; \\ 0 & \text{otherwise.} \end{cases} \tag{2}$$

This transformation achieves two things. Firstly it normalises the weights to be in the range 0 to 1, instead of the range 0 to 3 that the original ratings occupied. Secondly it normalises the strength with which each input node is connected to a target node with respect to the sum of all other inputs to the target. This means that multiple inputs to a target node cannot activate the target any more than a single input.

## 4.2 Parametric Propagation Schemes

We can now define how inference proceeds in the two parametric models: the linear and the noisy-OR models. Let $m$ denote the feature being counterfactually manipulated ('has wings' in our example), and let $A$ be a matrix in which each component $a_{i,t}$ represents the amount the model predicts feature $i$ to have changed as a result of the counterfactual

modification to $m$, after $t$ iterations. To initialise both models all predicted levels of change for features other than the manipulated feature, $m$, are initialised to 0:

$$a_{x:x\neq m,0} = 0 \qquad (3)$$

### 4.2.1 Linear Model

The update rules for each iteration of the linear model are defined as follows. The manipulated feature $m$ is set to an initial activation level of 1, indicating it has been counterfactually modified[1]. All other features have their activations set as specified below:

$$a_{x,t+1} = \sum_y c_{yx} a_{y,t} \qquad (4)$$

This condition states that a feature is expected to change in proportion to the degree to which the features that influence it have changed, given the initial alteration made to the manipulated feature $m$, and the degree to which they affect it. The general robustness of linear models of human judgements (Dawes, 1979) provides grounds for expecting a good correlation between the linear model and human counterfactual judgements.

### 4.2.2 Noisy-OR Model

The second model uses the noisy-OR gate (Pearl, 1988) to describe the propagation of information in causal inference. The noisy-OR gate assumes that each cause has an independent probability of failing to produce the effect, and that the effect will only be absent if all its associated causes fail to produce it. In the counterfactual model noisy-OR propagation is therefore formalised as follows:

$$a_{x,t+1} = 1 - \prod_y (1 - c_{yx} a_{y,t}) \qquad (5)$$

The questions people were asked to validate the two models measured how strongly they would believe in different features of a concept, if a specific feature was subtracted. This can be interpreted as the degree to which their belief in the target feature would vary given the presence and the absence of the manipulated feature. Accordingly, the output of the noisy-OR model was the difference in activation of each node when the manipulated node $m$ was set to 1 and 0 respectively[2].

### 4.2.3 Clamping

Because of the existence of loops in the dependency networks, if the counterfactually manipulated node is not clamped to its initial value activation can feed back through the network and change this value. This is likely to be undesirable, because it will mean the network will converge to a state in which the required counterfactual manipulation has not been successfully maintained, and that therefore its consequences have not been properly assimilated. The empirical performance of the two models was therefore considered when

the activation of the manipulated node was clamped to its initial value, and not clamped. The clamping constraint bears a close similarity to Pearl's (2000) '$do(X = x)$' operator, which prevents causes of a random variable $X$ affecting its value when an intervention has occurred in order to bring $X = x$ about.

### 4.2.4 Convergence

Propagation continues in both models until the activations for the features converge:

$$(\forall x)(|a_{x,t} - a_{x,t-1}| < \epsilon) \tag{6}$$

The models thus offer a spreading activation account of the changes induced in a conceptual network as a result of a counterfactual manipulation, their iterative nature allowing the effect of non-local influences to be accommodated.

## 5  Testing the Models

In order to test the validity of the two models we empirically studied people's intuitions about how they would expect concepts to change if they no longer possessed characteristic features. For example, participants were asked to imagine that robins did not in fact have wings. They were then asked to rate how strongly they agreed or disagreed with statements such as 'If robins didn't have wings, they would still be able to fly'. The task clearly requires participants to engage in counterfactual reasoning: robins do in fact have wings – in normal contexts at least – so participants are required to modify their standard conceptual representation in order to find out how this affects their belief in the other aspects of robins.

### 5.1  Method

Three features were chosen from each of the four concepts for which dependency information was available. These features were selected as having low, medium and high levels of centrality, as reported by Sloman, Love and Ahn (1998, Study 1). This was to ensure that counterfactuals revolving around more and less important features of a concept were considered in the study.

Each selected feature formed the basis of a counterfactual manipulation. For example, if the concept was robin and the selected feature was 'has wings', then subjects were asked to imagine that robins didn't have wings. Participants were then asked how strongly they believed that the concept in question would still possess each of its remaining features if it no longer possessed the selected feature. For example, they would read 'If robins didn't have wings, they would still be able to fly' and be asked to rate how strongly they agreed with it.

Ratings were elicited on a 1-7 point scale anchored by 'strongly disagree' at the lower end and 'strongly agree' at the upper end. The ratings provided by participants can be regarded as estimates of how much people expect the features of a concept to change if the concept were counterfactually modified in the specified way. If the models are good ones we would therefore expect there to be a correlation between their predictions and the judgements of the participants.

### 5.2  Design and Materials

Participants were randomly presented with 4 of the 12 counterfactual manipulations, and were asked to rate their agreement with counterfactual statements about the remaining,

| Counterfactual Concept | n | Linear Model | | Noisy-OR Model | |
|---|---|---|---|---|---|
| | | Clamped | Non-Clamped | Clamped | Non-Clamped |
| robin-wings | 13 | -0.870** | -0.044 | -0.739** | -0.062 |
| robin-lays-eggs | 13 | -0.521* | -0.105 | -0.278 | 0.121 |
| robin-eats-worms | 13 | -0.066 | -0.069 | -0.009 | -0.017 |
| chair-back | 8 | -0.451 | 0.191 | -0.178 | 0.148 |
| chair-arms | 8 | -0.530 | 0.042 | -0.358 | 0.036 |
| chair-holds-people | 8 | -0.815** | -0.928** | -0.917** | -0.957** |
| guitar-neck | 8 | -0.760* | -0.242 | -0.381 | -0.181 |
| guitar-makes-sound | 8 | -0.889** | -0.920** | -0.939** | 0.895** |
| guitar-used-by-music-groups | 8 | 0.235 | 0.225 | 0.290 | 0.263 |
| apple-grows-on-trees | 8 | -0.748* | -0.838** | -0.905** | -0.921** |
| apple-edible | 8 | -0.207 | 0.361 | -0.288 | 0.000 |
| apple-stem | 8 | -0.965** | -0.948** | -0.961** | -0.893** |
| Mean | | **-0.549** | **-0.273** | **-0.472** | **-0.131** |

Table 1: The correlation between the linear and noisy-OR models, in the clamped and non-clamped conditions, with participants' empirical judgements about corresponding inferences. All comparisons were one-tailed (* $p < 0.05$; ** $p < 0.01$).

unmanipulated features of the concept. People read an introductory passage for each inference in which they were asked to 'Imagine that robins didn't have wings. If this was true, how much would you agree or disagree with the following statements...' They were then asked to rate their agreement with the specific inferences.

### 5.3 Participants

38 members of the Division of Informatics, University of Edinburgh, took part in the study. All participants were volunteers, and no reward was offered for participation.

### 5.4 Results

The correlation of the two models, in the clamped and non-clamped conditions, is shown in Table 1. A $2 \times 2$ repeated-measures ANOVA revealed that there was a main effect of clamping ($F(1, 11) = 10.56$, $p = 0.008$), no main effect of propagation method ($F(1, 11) = 1.90$, $p = 0.196$), and no interaction effect ($F(1, 11) < 1$). The correlations of both the linear (Wilcoxon Test, Z = 2.82, $p < 0.0025$, one-tailed) and the noisy-OR model (Wilcoxon Test, Z = 2.67, $p < 0.01$, one-tailed) differed significantly from 0 when clamping was used.

### 5.5 Discussion

The simulation results show that clamping is necessary to the success of the counterfactual models; this thus constitutes an empirical validation of Pearl's use of the '$do(X = x)$' operator in modelling counterfactuals. In addition, both the models capture the empirical patterns with some degree of success, so further work is required to tease them apart.

## 6 Exploring Non-Parametric Approaches

The models of counterfactual reasoning we have presented both make parametric assumptions. Although non-parametric models in general offer greater flexibility, there are two main reasons – one logistical and one methodological – why applying them in this context may be problematic.

### 6.1  A Logistical Reason: Conditional Probability Tables

Bayesian Belief Networks (BBNs) define conditional dependence relations in terms of graph structures like the dependency structures used by the present model. This makes them an obvious choice of normative model for counterfactual inference. However, there are certain problems that make the application of a non-parametric BBN to counterfactual reasoning problematic.

For non-parametric inference a joint conditional probability table needs to be defined for *all* the variables upon which a target node is conditioned. In other words, it's not sufficient to know $p(e|c_1)$, $p(e|c_2)$, ..., $p(e|c_n)$ alone; instead, $p(e|c_1, ...c_n)$ is required. This leads to a combinatorial explosion in the number of parameters required. If $V$ is a vector of $i$ elements in which $v_i$ represents the number of discrete classes that the random variable $C_i$ can take, then the number of conditional probabilities required to compute the interaction between $C_{1-n}$ and $E$ in the general case is:

$$\prod_{i=1}^{n} v_i \tag{7}$$

On the assumption that features can normally be represented by two classes (present or absent), the number of probability judgements required to successfully apply a non-parametric BBN to all four of Sloman, Love and Ahn's (1998) concepts is 3888. Aside from the obvious logistical difficulties in obtaining estimates of this number of parameters from people, attribution theorists suggest that simplifying assumptions are often made in causal inference (Kelley, 1972). If this is the case then it should be possible to specify a parametric model which appropriately captures these patterns, as we have attempted to do with the models in this paper, thus obviating the need for a fully general non-parametric approach.

### 6.2  A Methodological Reason: Patterns of Interaction

Parametric models are special cases of non-parametric models: this means that a non-parametric model will be able to capture patterns of interaction between causes that a parametric model may be unable to express. A risk concomitant with the generality of non-parametric models is that they can gloss over important limitations in human inference. Although a non-parametric approach, with exhaustively estimated conditional probability parameters, would likely fit people's counterfactual judgements satisfactorily, it would not inform us about the limitations in our ability to process causal interactions. A parametric approach, however, allows one to adopt an incremental approach to modelling in which such limitations can be made explicit: parametric models can be generalised when there is empirical evidence that they fail to capture a particular kind of interaction. Parametric approaches go hand-in-hand, then, with an empirical investigation of our treatment of causal interactions. Obtaining a good fit with data is not of sole importance in cognitive modelling: it is also important for the model to make explicit the assumptions it is predicated on, and parametric approaches allow this to be done, hopefully making causal principles explicit which would otherwise lie latent in an exhaustive conditional probability table.

## 7  Closing Thoughts

Given the lack of quantitative models of counterfactual reasoning, we believe the models we have presented in this paper constitute a significant contribution to our understanding of this process. Notably, the models achieved a significant correlation across a sizeable data-set (111 data-points), with *no* free parameters. However, there are limitations to the

current models. As stated, the models both assume that causal factors contribute independently to a target factor, and this is clearly not always the case. Although a non-parametric Bayesian model with an exhaustive conditional probability table could accommodate all possible interaction effects between causal factors, as argued in the previous section, this would not necessarily be all that enlightening. It is up to further empirical work to unearth the principles underpinning our processing of causal interactions (e.g., Kelley, 1972); these principles can then be made explicit in future parametric models to yield a fuller understanding of human inference. In the future we intend to examine our treatment of causal interactions empirically, in order to reach a better understanding of the appropriate way to model counterfactual reasoning.

## Acknowledgements

We would like to thank Tom Griffiths, Brad Love, Steven Sloman and Josh Tenenbaum for their discussion of the ideas presented in this paper.

## Footnotes

[1]Note that the performance of the linear model does not depend crucially on the activation of $m$ being set to 1, as solutions for $A$ at convergence are simply multiples of the initial value selected and hence will not affect the correlational results.

[2]This highlights an interesting difference in the output of the two models: the linear model outputs the degree to which a feature is expected to change as a result of a counterfactual manipulation directly, whereas the noisy-OR model outputs probabilities which need to be converted into an expected degree of change (expressed in Pearl's causal calculus as $P(f_i|do[M = 1]) - P(f_i|do[M = 0])$).

## References

[1] Byrne R.M.J. and Tasso A. (1999). Counterfactual Reasoning with Factual, Possible, and Counterfactual Conditionals, *Memory & Cognition*, **27(4)**, 726-740.

[2] Dawes R.M. (1979). The Robust Beauty of Improper Linear Models in Decision Making, *American Psychologist*, **34**, 571-582.

[3] Goodman N. (1983; 4th edition). *Fact, Fiction, and Forecast*, Harvard University Press, Cambridge, Massachusetts.

[4] Griffiths T. (August 2001). Assessing Interventions in Linear Belief Networks.

[5] Kahneman D. and Miller D.T. (1986). Norm Theory: Comparing Reality to its Alternatives, *Psychological Review*, **93(2)**, 136-153.

[6] Kahneman D., Slovic P. and Tversky A. (1982; eds.). *Judgment Under Uncertainty: Heuristics and Biases*, Cambridge University Press, Cambridge, UK.

[7] Kelley H.H. (1972). Causal Schemata and the Attribution Process. In Jones, Kanouse, Kelley, Nisbett, Valins and Weiners (eds.), *Attribution: Perceiving the Causes of Behavior*, Chapter 9, 151-174, General Learning Press, Morristown, New Jersey.

[8] Lewis D.K. (1973). *Counterfactuals*, Harvard University Press, Cambridge, Massachusetts.

[9] Murphy K.P., Weiss Y. and Jordan M.I. (1999). Loopy Belief Propagation for Approximate Inference: An Empirical Study, *Proceedings of Uncertainty in AI*, 467-475.

[10] Pearl J. (1988). *Probabilistic Reasoning in Intelligent Systems: Networks of Plausible Inference*, Morgan Kaufmann, San Mateo, California.

[11] Pearl J. (2000). *Causality: Models, Reasoning, and Inference*, Cambridge University Press, Cambridge.

[12] Roese N.J. (1997). Counterfactual Thinking, *Psychological Bulletin*, **121**, 133-148.

[13] Sloman S., Love B.C. and Ahn W.K. (1998). Feature Centrality and Conceptual Coherence, *Cognitive Science*, **22(2)**, 189-228.

[14] Yarlett D.G. and Ramscar M.J.A. (2001). Structural Determinants of Counterfactual Reasoning, *Proceedings of the 23rd Annual Conference of the Cognitive Science Society*, 1154-1159.

[15] Yarlett D.G. and Ramscar M.J.A. (in preparation). Uncertainty in Causal and Counterfactual Inference.
